# Neighbourhood Components Analysis

**Jacob Goldberger,  Sam Roweis,  Geoff Hinton,  Ruslan Salakhutdinov**
Department of Computer Science, University of Toronto
{jacob,roweis,hinton,rsalakhu}@cs.toronto.edu

## Abstract

In this paper we propose a novel method for learning a Mahalanobis distance measure to be used in the KNN classification algorithm. The algorithm directly maximizes a stochastic variant of the leave-one-out KNN score on the training set. It can also learn a low-dimensional linear embedding of labeled data that can be used for data visualization and fast classification. Unlike other methods, our classification model is non-parametric, making no assumptions about the shape of the class distributions or the boundaries between them. The performance of the method is demonstrated on several data sets, both for metric learning and linear dimensionality reduction.

## 1   Introduction

Nearest neighbor (KNN) is an extremely simple yet surprisingly effective method for classification. Its appeal stems from the fact that its decision surfaces are nonlinear, there is only a single integer parameter (which is easily tuned with cross-validation), and the expected quality of predictions improves automatically as the amount of training data increases. These advantages, shared by many non-parametric methods, reflect the fact that although the final classification machine has quite high capacity (since it accesses the entire reservoir of training data at test time), the trivial learning procedure rarely causes overfitting itself.

However, KNN suffers from two very serious drawbacks. The first is computational, since it must store and search through the entire training set in order to classify a single test point. (Storage can potentially be reduced by "editing" or "thinning" the training data; and in low dimensional input spaces, the search problem can be mitigated by employing data structures such as KD-trees or ball-trees[4].) The second is a modeling issue: how should the distance metric used to define the "nearest" neighbours of a test point be defined? In this paper, we attack both of these difficulties by learning a quadratic distance metric which optimizes the expected leave-one-out classification error on the training data when used with a stochastic neighbour selection rule. Furthermore, we can force the learned distance metric to be low rank, thus substantially reducing storage and search costs at test time.

## 2   Stochastic Nearest Neighbours for Distance Metric Learning

We begin with a labeled data set consisting of $n$ real-valued input vectors $x_1, \ldots, x_n$ in $\mathcal{R}^D$ and corresponding class labels $c_1, \ldots, c_n$. We want to find a distance metric that maximizes

the performance of nearest neighbour classification. Ideally, we would like to optimize performance on future test data, but since we do not know the true data distribution we instead attempt to optimize leave-one-out (LOO) performance on the training data.

In what follows, we restrict ourselves to learning Mahalanobis (quadratic) distance metrics, which can always be represented by symmetric positive semi-definite matrices. We estimate such metrics through their inverse square roots, by learning a *linear transformation of the input space such that in the transformed space, KNN performs well*. If we denote the transformation by a matrix $A$ we are effectively learning a metric $Q = A^\top A$ such that $d(x, y) = (x - y)^\top Q(x - y) = (Ax - Ay)^\top(Ax - Ay)$.

The actual leave-one-out classification error of KNN is quite a discontinuous function of the transformation $A$, since an infinitesimal change in $A$ may change the neighbour graph and thus affect LOO classification performance by a finite amount. Instead, we adopt a more well behaved measure of nearest neighbour performance, by introducing a differentiable cost function based on stochastic ("soft") neighbour assignments in the transformed space. In particular, each point $i$ selects another point $j$ as its neighbour with some probability $p_{ij}$, and inherits its class label from the point it selects. We define the $p_{ij}$ using a softmax over Euclidean distances in the transformed space:

$$p_{ij} = \frac{\exp(-\|Ax_i - Ax_j\|^2)}{\sum_{k \neq i} \exp(-\|Ax_i - Ax_k\|^2)} \qquad , \qquad p_{ii} = 0 \qquad (1)$$

Under this stochastic selection rule, we can compute the probability $p_i$ that point $i$ will be correctly classified (denote the set of points in the same class as $i$ by $C_i = \{j | c_i = c_j\}$):

$$p_i = \sum_{j \in C_i} p_{ij} \qquad (2)$$

The objective we maximize is the *expected number of points correctly classified* under this scheme:

$$f(A) = \sum_i \sum_{j \in C_i} p_{ij} = \sum_i p_i \qquad (3)$$

Differentiating $f$ with respect to the transformation matrix $A$ yields a gradient rule which we can use for learning (denote $x_{ij} = x_i - x_j$):

$$\frac{\partial f}{\partial A} = -2A \sum_i \sum_{j \in C_i} p_{ij}(x_{ij}x_{ij}^\top - \sum_k p_{ik}x_{ik}x_{ik}^\top) \qquad (4)$$

Reordering the terms we obtain a more efficiently computed expression:

$$\frac{\partial f}{\partial A} = 2A \sum_i \left( p_i \sum_k p_{ik}x_{ik}x_{ik}^\top - \sum_{j \in C_i} p_{ij}x_{ij}x_{ij}^\top \right) \qquad (5)$$

Our algorithm – which we dub Neighbourhood Components Analysis (NCA)– is extremely simple: maximize the above objective (3) using a gradient based optimizer such as delta-bar-delta or conjugate gradients. Of course, since the cost function above is not convex, some care must be taken to avoid local maxima during training. However, unlike many other objective functions (where good optima are not necessarily deep but rather broad) it has been our experience that the larger we can drive $f$ during training the better our test performance will be. In other words, we have never observed an "overtraining" effect.

Notice that by learning the overall scale of $A$ as well as the relative directions of its rows we are also effectively learning a real-valued estimate of the optimal number of neighbours (K). This estimate appears as the effective perplexity of the distributions $p_{ij}$. If the learning

procedure wants to reduce the effective perplexity (consult fewer neighbours) it can scale up $A$ uniformly; similarly by scaling down all the entries in $A$ it can increase the perplexity of and effectively average over more neighbours during the stochastic selection.

Maximizing the objective function $f(A)$ is equivalent to minimizing the $L_1$ norm between the true class distribution (having probability one on the true class) and the stochastic class distribution induced by $p_{ij}$ via $A$. A natural alternative distance is the KL-divergence which induces the following objective function:

$$g(A) = \sum_i \log(\sum_{j \in C_i} p_{ij}) = \sum_i \log(p_i) \tag{6}$$

Maximizing this objective would correspond to maximizing the probability of obtaining a *perfect (error free) classification of the entire training set*. The gradient of $g(A)$ is even simpler than that of $f(A)$:

$$\frac{\partial g}{\partial A} = 2A \sum_i \left( \sum_k p_{ik} x_{ik} x_{ik}^\top - \frac{\sum_{j \in C_i} p_{ij} x_{ij} x_{ij}^\top}{\sum_{j \in C_i} p_{ij}} \right) \tag{7}$$

We have experimented with optimizing this cost function as well, and found both the transformations learned and the performance results on training and testing data to be very similar to those obtained with the original cost function.

To speed up the gradient computation, the sums that appear in equations (5) and (7) over the data points and over the neigbours of each point, can be truncated (one because we can do stochastic gradient rather than exact gradient and the other because $p_{ij}$ drops off quickly).

## 3 Low Rank Distance Metrics and Nonsquare Projection

Often it is useful to reduce the dimensionality of input data, either for computational savings or for regularization of a subsequent learning algorithm. Linear dimensionality reduction techniques (which apply a linear operator to the original data in order to arrive at the reduced representation) are popular because they are both fast and themselves relatively immune to overfitting. Because they implement only affine maps, linear projections also preserve some essential topology of the original data. Many approaches exist for linear dimensionality reduction, ranging from purely unsupervised approaches (such as factor analysis, principal components analysis and independent components analysis) to methods which make use of class labels in addition to input features such as linear discriminant analysis (LDA)[3] possibly combined with relevant components analysis (RCA)[1].

By restricting $A$ to be a nonsquare matrix of size $d \times D$, NCA can also do linear dimensionality reduction. In this case, the learned metric will be low rank, and the transformed inputs will lie in $\mathcal{R}^d$. (Since the transformation is linear, without loss of generality we only consider the case $d \leq D$. ) By making such a restriction, we can potentially reap many further benefits beyond the already convenient method for learning a KNN distance metric. In particular, by choosing $d \ll D$ we can vastly reduce the storage and search-time requirements of KNN. Selecting $d = 2$ or $d = 3$ we can also compute useful low dimensional visualizations on labeled datasets, using only a linear projection. The algorithm is exactly the same: optimize the cost function (3) using gradient descent on a nonsquare $A$. Our method requires no matrix inversions and assumes no parametric model (Gaussian or otherwise) for the class distributions or the boundaries between them. For now, the dimensionality of the reduced representation (the number of rows in $A$) must be set by the user.

By using an highly rectangular $A$ so that $d \ll D$, we can significantly reduce the computational load of KNN at the expense of restricting the allowable metrics to be those of

rank at most $d$. To achieve this, we apply the NCA learning algorithm to find the optimal transformation $A$, and then we store only the projections of the training points $y_n = Ax_n$ (as well as their labels). At test time, we classify a new point $x_{test}$ by first computing its projection $y_{test} = Ax_{test}$ and then doing KNN classification on $y_{test}$ using the $y_n$ and a simple Euclidean metric. If $d$ is relatively small (say less than 10), we can preprocess the $y_n$ by building a KD-tree or a ball-tree to further increase the speed of search at test time. The storage requirements of this method are $O(dN) + Dd$ compared with $O(DN)$ for KNN in the original input space.

## 4 Experiments in Metric Learning and Dimensionality Reduction

We have evaluated the NCA algorithm against standard distance metrics for KNN and other methods for linear dimensionality reduction. In our experiments, we have used 6 data sets (5 from the UC Irvine repository). We compared the NCA transformation obtained from optimizing $f$ (for square $A$) on the training set with the default Euclidean distance $A = I$, the "whitening" transformation , $A = \Sigma^{-\frac{1}{2}}$ (where $\Sigma$ is the sample data covariance matrix), and the RCA [1] transformation $A = \Sigma_w^{-\frac{1}{2}}$ (where $\Sigma_w$ is the average of the within-class covariance matrices). We also investigated the behaviour of NCA when $A$ is restricted to be diagonal, allowing only axis aligned Mahalanobis measures.

Figure 1 shows that the training and (more importantly) testing performance of NCA is consistently the same as or better than that of other Mahalanobis distance measures for KNN, despite the relative simplicity of the NCA objective function and the fact that the distance metric being learned is nothing more than a positive definite matrix $A^\top A$.

We have also investigated the use of linear dimensionality reduction using NCA (with non-square $A$) for visualization as well as reduced-complexity classification on several datasets. In figure 2 we show 4 examples of 2-D visualization. First, we generated a synthetic three-dimensional dataset (shown in top row of figure 2) which consists of 5 classes (shown by different colors). In two dimensions, the classes are distributed in concentric circles, while the third dimension is just Gaussian noise, uncorrelated with the other dimensions or the class label. If the noise variance is large enough, the projection found by PCA is forced to include the noise (as shown on the top left of figure 2). (A full rank Euclidean metric would also be misled by this dimension.) The classes are not convex and cannot be linearly separated, hence the results obtained from LDA will be inappropriate (as shown in figure 2). In contrast, NCA adaptively finds the best projection without assuming any parametric structure in the low dimensional representation. We have also applied NCA to the UCI "wine" dataset, which consists of 178 points labeled into 3 classes and to a database of gray-scale images of faces consisting of 18 classes (each a separate individual) and 560 dimensions (image size is $20 \times 28$). The face dataset consists of 1800 images (100 for each person). Finally, we applied our algorithm to a subset of the USPS dataset of handwritten digit images, consisting of the first five digit classes ("one" through "five"). The grayscale images were downsampled to $8 \times 8$ pixel resolution resulting in 64 dimensions.

As can be seen in figure 2 when a two-dimensional projection is used, the classes are consistently much better separated by the NCA transformation than by either PCA (which is unsupervised) or LDA (which has access to the class labels). Of course, the NCA transformation is still only a linear projection, just optimized with a cost function which explicitly encourages local separation. To further quantify the projection results we can apply a nearest-neighbor classification in the projected space. Using the same projection learned at training time, we project the training set and all future test points and perform KNN in the low-dimensional space using the Euclidean measure. The results under the PCA, LDA, LDA followed by RCA and NCA transformations (using K=1) appear in figure 1. The NCA projection consistently gives superior performance in this highly constrained low-

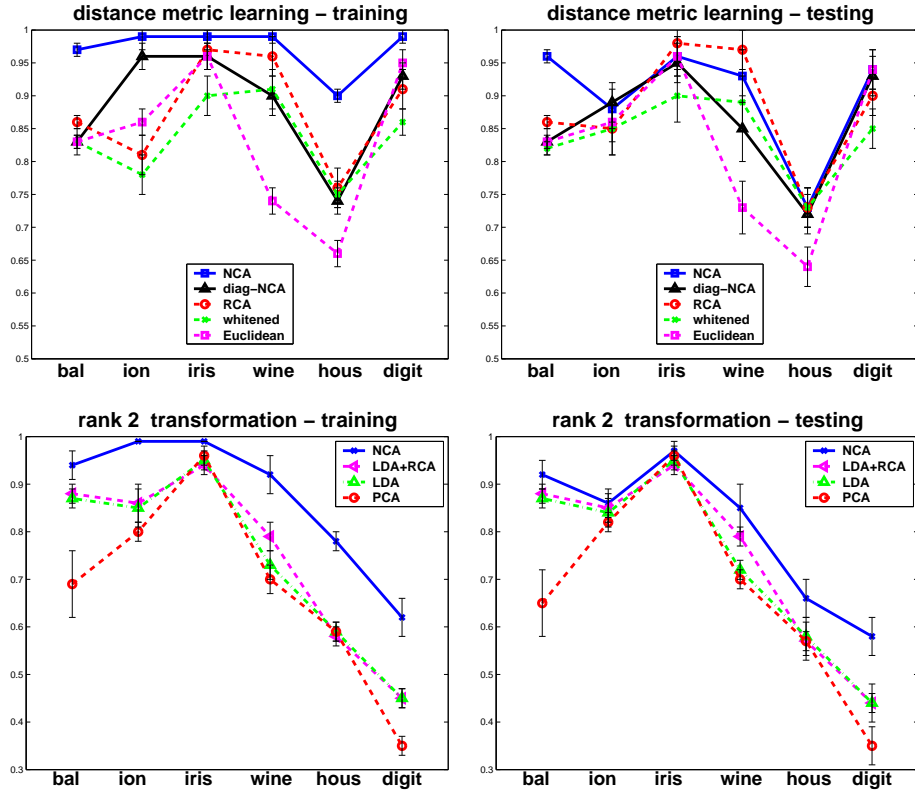

Figure 1: KNN classification accuracy (left train, right test) on UCI datasets balance, ionosphere, iris, wine and housing and on the USPS handwritten digits. Results are averages over 40 realizations of splitting each dataset into training (70%) and testing (30%) subsets (for USPS 200 images for each of the 10 digit classes were used for training and 500 for testing). Top panels show distance metric learning (square $A$) and bottom panels show linear dimensionality reduction down to $d = 2$.

rank KNN setting. In summary, we have found that when labeled data is available, NCA performs better both in terms of classification performance in the projected representation and in terms of visualization of class separation as compared to the standard methods of PCA and LDA.

## 5 Extensions to Continuous Labels and Semi-Supervised Learning

Although we have focused here on discrete classes, linear transformations and fully supervised learning, many extensions of this basic idea are possible. Clearly, a nonlinear transformation function $A(\cdot)$ could be learned using any architecture (such as a multilayer perceptron) trainable by gradient methods. Furthermore, it is possible to extend the classification framework presented above to the case of a real valued (continuous) supervision signal by defining the set of "correct matches" $C_i$ for point $i$ to be those points $j$ having similar (continuous) targets. This naturally leads to the idea of "soft matches", in which the objective function becomes a sum over all pairs, each weighted by their agreement according to the targets. Learning under such an objective can still proceed even in settings where the targets are not explicitly provided as long as information identifying close pairs

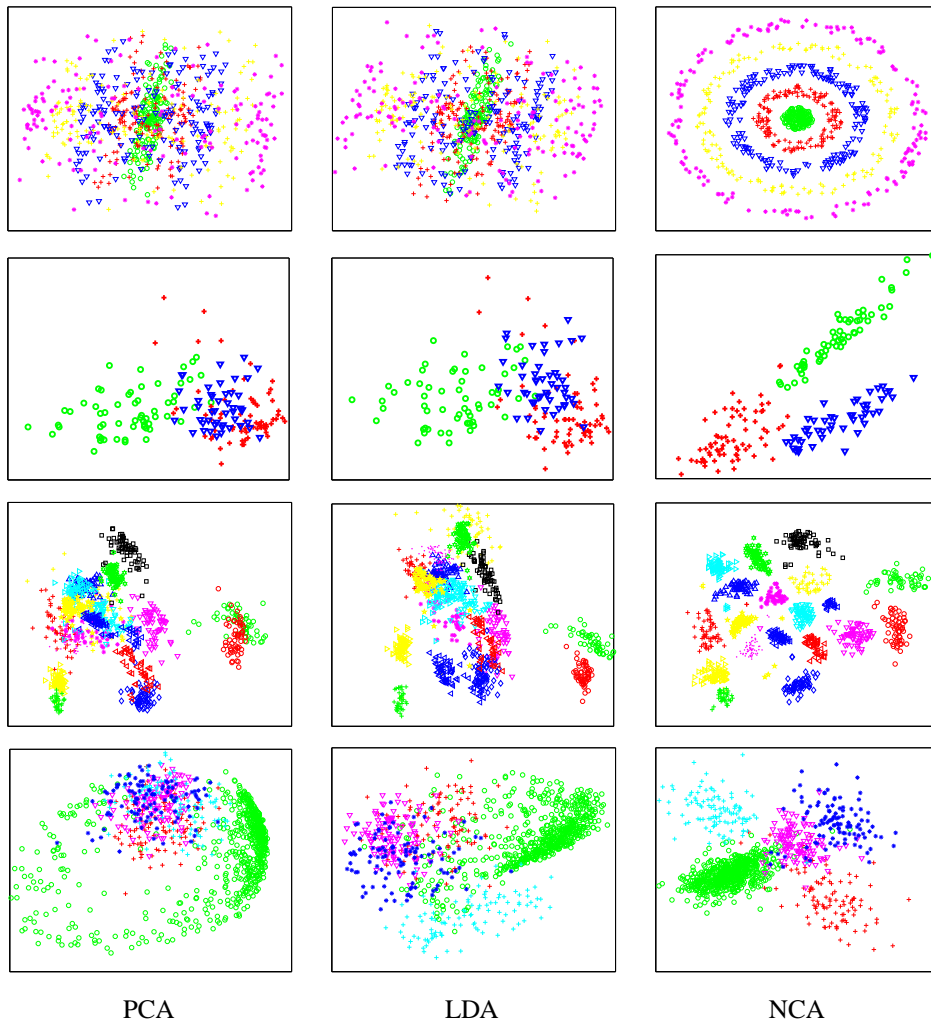

PCA                    LDA                    NCA

Figure 2: Dataset visualization results of PCA, LDA and NCA applied to (from top) the "concentric rings", "wine", "faces" and "digits" datasets. The data are reduced from their original dimensionalities (D=3,D=13,D=560,D=256 respectively) to the d=2 dimensions show.

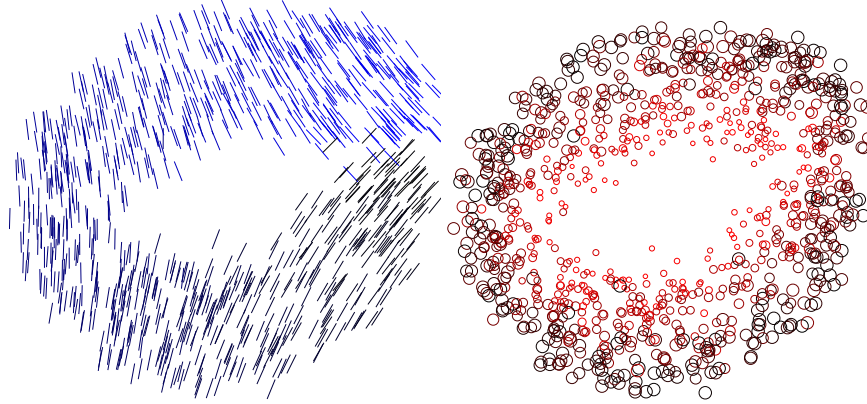

Figure 3: The two dimensional outputs of the neural network on a set of test cases. On the left, each point is shown using a line segment that has the same orientation as the input face. On the right, the same points are shown again with the size of the circle representing the size of the face.

is available. Such semi-supervised tasks often arise in domains with strong spatial or temporal continuity constraints on the supervision, e.g. in a video of a person's face we may assume that pose, and expression vary slowly in time even if no individual frames are ever labeled explicitly with numerical pose or expression values.

To illustrate this, we generate pairs of faces in the following way: First we choose two faces at random from the FERET-B dataset (5000 isolated faces that have a standard orientation and scale). The first face is rotated by an angle uniformly distributed between $\pm 45^o$ and scaled to have a height uniformly distributed between 25 and 35 pixels. The second face (which is of a different person) is given the same rotation and scaling but with Gaussian noise of $\pm 1.22^o$ and $\pm 1.5$ pixels. The pair is given a weight, $w_{ab}$, which is the probability density of the added noise divided by its maximum possible value. We then trained a neural network with one hidden layer of 100 logistic units to map from the $35 \times 35$ pixel intensities of a face to a point, $y$, in a 2-D output space. Backpropagation was used to minimize the cost function in Eq. 8 which encourages the faces in a pair to be placed close together:

$$Cost = - \sum_{pair(a,b)} w_{ab} \log \left( \frac{\exp(-||y_a - y_b||^2)}{\sum_{c,d} \exp(-||y_c - y_d||^2)} \right) \quad (8)$$

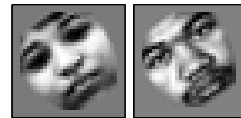
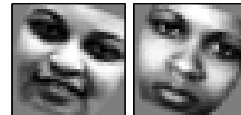

where $c$ and $d$ are indices over all of the faces, not just the ones that form a pair. Four example faces are shown to the right; horizontally the pairs agree and vertically they do not. Figure 3 above shows that the feedforward neural network discovered polar coordinates without the user having to decide how to represent scale and orientation in the output space.

## 6    Relationships to Other Methods and Conclusions

Several papers recently addressed the problem of learning Mahalanobis distance functions given labeled data or at least side-information of the form of equivalence constraints. Two related methods are RCA [1] and a convex optimization based algorithm [7]. RCA is implicitly assuming a Gaussian distribution for each class (so it can be described using only the first two moments of the class-conditional distribution). Xing et. al attempt to find a transformation which minimizes all pairwise squared distances between points in the

same class; this implicitly assumes that classes form a single compact connected set. For highly multimodal class distributions this cost function will be severely penalized. Lowe[6] proposed a method similar to ours but used a more limited idea for learning a nearest neighbour distance metric. In his approach, the metric is constrained to be diagonal (as well, it is somewhat redundantly parameterized), and the objective function corresponds to the average *squared error* between the true class distribution and the predicted distribution, which is not entirely appropriate in a more probabilistic setting.

In parallel there has been work on learning low rank transformations for fast classification and visualization. The classic LDA algorithm[3] is optimal if all class distributions are Gaussian with a single shared covariance; this assumption, however is rarely true. LDA also suffers from a small sample size problem when dealing with high-dimensional data when the within-class scatter matrix is nearly singular[2]. Recent variants of LDA (e.g. [5], [2]) make the transformation more robust to outliers and to numerical instability when not enough datapoints are available. (This problem does not exist in our method since there is no need for a matrix inversion.)

In general, there are two classes of regularization assumption that are common in linear methods for classification. The first is a strong parametric assumption about the structure of the class distributions (typically enforcing connected or even convex structure); the second is an assumption about the decision boundary (typically enforcing a hyperplane). Our method makes neither of these assumptions, relying instead on the strong regularization imposed by restricting ourselves to a linear transformation of the original inputs.

Future research on the NCA model will investigate using local estimates of $K$ as derived from the entropy of the distributions $p_{ij}$; the possible use of a stochastic classification rule at test time; and more systematic comparisons between the objective functions $f$ and $g$.

To conclude, we have introduced a novel non-parametric learning method — NCA — that handles the tasks of distance learning and dimensionality reduction in a unified manner. Although much recent effort has focused on non-linear methods, we feel that linear embedding has still not fully fulfilled its potential for either visualization or learning.

### Acknowledgments

Thanks to David Heckerman and Paul Viola for suggesting that we investigate the alternative cost $g(A)$ and the case of diagonal $A$.

## References

[1] A. Bar-Hillel, T. Hertz, N. Shental, and D. Weinshall. Learning distance functions using equivalence relation. In *International Conference on Machine Learning*, 2003.

[2] L. Chen, H. Liao, M. Ko, J. Lin, and G. Yu. A new lda-based face recognition system which can solve the small sample size problem. In *Pattern Recognition*, pages 1713–1726, 2000.

[3] R. A. Fisher. The use of multiple measurements in taxonomic problems. In *Annual of Eugenic*, pages 179–188, 1936.

[4] J. Friedman, J.bentley, and R. Finkel. An algorithm for finding best matches in logarithmic expected time. In *ACM*, 1977.

[5] Y. Koren and L. Carmel. Robust linear dimensionality reduction. In *IEEE Trans. Vis. and Comp. Graph.*, pages 459–470, 2004.

[6] D. Lowe. Similarity metric learning for a variable kernel classifier. In *Neural Computation*, pages 72–85, 1995.

[7] E.P. Xing, A. Y. Ng, M.I. Jordan, and S. Russell. Distance learning metric. In *Proc. of Neural Information Processing Systems*, 2003.
